# The Error Coding and Substitution PaCTs

GARETH JAMES
and
TREVOR HASTIE
Department of Statistics, Stanford University

## Abstract

A new class of *plug in* classification techniques have recently been developed in the statistics and machine learning literature. A plug in classification technique (PaCT) is a method that takes a standard classifier (such as LDA or TREES) and plugs it into an algorithm to produce a new classifier. The standard classifier is known as the *Plug in Classifier* (PiC). These methods often produce large improvements over using a single classifier. In this paper we investigate one of these methods and give some motivation for its success.

## 1  Introduction

Dietterich and Bakiri (1995) suggested the following method, motivated by Error Correcting Coding Theory, for solving $k$ class classification problems using binary classifiers.

- Produce a $k$ by $B$ ($B$ large) binary coding matrix, ie a matrix of zeros and ones. We will denote this matrix by $Z$, its $i,j$th component by $Z_{ij}$, its $i$th row by $\mathbf{Z}_i$ and its $j$th column by $\mathbf{Z}^j$.

- Use the first column of the coding matrix ($\mathbf{Z}^1$) to create two *super* groups by assigning all groups with a one in the corresponding element of $\mathbf{Z}^1$ to super group one and all other groups to super group zero.

- Train your plug in classifier (PiC) on the new two class problem.

- Repeat the process for each of the $B$ columns ($\mathbf{Z}^1, \mathbf{Z}^2, \dots, \mathbf{Z}^B$) to produce $B$ trained classifiers.

- For a new test point apply each of the $B$ classifiers to it. Each classifier will produce a $\hat{p}_j$ which is the estimated probability the test point comes from the $j$th super group one. This will produce a vector of probability estimates, $\hat{\mathbf{p}} = (\hat{p}_1, \hat{p}_2, \dots, \hat{p}_B)^T$.

- To classify the point calculate $L_i = \sum_{j=1}^{B} |\hat{p}_j - Z_{ij}|$ for each of the $k$ groups (ie for $i$ from 1 to $k$). This is the L1 distance between $\hat{\mathbf{p}}$ and $\mathbf{Z}_i$ (the $i$th row of $Z$). Classify to the group with lowest L1 distance or equivalently $\arg_i \min L_i$

We call this the ECOC PaCT. Each row in the coding matrix corresponds to a unique (non-minimal) coding for the appropriate class. Dietterich's motivation was that this allowed *errors* in individual classifiers to be *corrected* so if a small number of classifiers gave a bad fit they did not unduly influence the final classification. Several PiC's have been tested. The best results were obtained by using *tree's*, so all the experiments in this paper are stated using a standard CART PiC. Note however, that the theorems are general to any PiC.

In the past it has been assumed that the improvements shown by this method were attributable to the error coding structure and much effort has been devoted to choosing an *optimal* coding matrix. In this paper we develop results which suggest that a randomized coding matrix should match (or exceed) the performance of a *designed* matrix.

## 2 The Coding Matrix

Empirical results (see Dietterich and Bakiri (1995)) suggest that the ECOC PaCT can produce large improvements over a standard $k$ class tree classifier. However, they do not shed any light on why this should be the case. To answer this question we need to explore its probability structure. The coding matrix, $Z$, is central to the PaCT. In the past the usual approach has been to choose one with as large a separation between rows ($\mathbf{Z}_i$) as possible (in terms of hamming distance) on the basis that this allows the largest number of *errors* to be corrected. In the next two sections we will examine the tradeoffs between a *designed (deterministic)* and a *completely randomized* matrix.

Some of the results that follow will make use of the following assumption.

$$E[\hat{p}_j \mid Z, X] = \sum_{i=1}^{k} Z_{ij}q_i = \mathbf{Z}^{j^T}\mathbf{q} \quad j = 1, \dots, B \tag{1}$$

where $q_i = P(G_i \mid X)$ is the posterior probability that the test observation is from group $i$ given that our predictor variable is $X$. This is an unbiasedness assumption. It states that on average our classifier will estimate the probability of being in super group one correctly. The assumption is probably not too bad given that trees are considered to have low bias.

### 2.1 Deterministic Coding Matrix

Let $\bar{D}_i = 1 - 2L_i/B$ for $i = 1 \dots k$. Notice that $\arg_i \min L_i = \arg_i \max \bar{D}_i$ so using $\bar{D}_i$ to classify is identical to the ECOC PaCT. Theorem 3 in section 2.2 explains why this is an intuitive transformation to use.

Obviously no PaCT can outperform the Bayes Classifier. However we would hope that it would achieve the Bayes Error Rate when we use the Bayes Classifier as our PiC for each 2 class problem. We have defined this property as Bayes Optimality. Bayes Optimality is essentially a consistency result. It states, if our PiC converges to the Bayes Classifier, as the training sample size increases, then so will the PaCT.

**Definition 1** *A PaCT is said to be Bayes Optimal if, for any test set, it always classifies to the bayes group when the Bayes Classifier is our PiC.*

For the ECOC PaCT this means that $\arg_i \max q_i = \arg_i \max \bar{D}_i$, for all points in the predictor space, when we use the Bayes Classifier as our PiC. However it can be shown that in this case

$$\bar{D}_i = 1 - \frac{2}{B}\sum_{l \neq i} q_l \sum_{j=1}^{B}(Z_{lj} - Z_{ij})^2 \quad i = 1, \dots, k$$

It is not clear from this expression why there should be any guarantee that $\arg_i \max \bar{D}_i = \arg_i \max q_i$. In fact the following theorem tells us that only in very restricted circumstances will the ECOC PaCT be Bayes Optimal.

**Theorem 1** *The Error Coding method is Bayes Optimal iff the Hamming distance between every pair of rows of the coding matrix is equal.*

The hamming distance between two binary vectors is the number of points where they differ. For general $B$ and $k$ there is no known way to generate a matrix with this property so the ECOC PaCT will not be Bayes Optimal.

## 2.2   Random Coding Matrix

We have seen in the previous section that there are potential problems with using a deterministic matrix. Now suppose we randomly generate a coding matrix by choosing a zero or one with equal probability for every coordinate. Let $\mu_i = E(1 - 2|\hat{p}_1 - Z_{i1}| \mid \mathcal{T})$ where $\mathcal{T}$ is the training set. Then $\mu_i$ is the conditional expectation of $\bar{D}_i$ and we can prove the following theorem.

**Theorem 2** *For a random coding matrix, conditional on $\mathcal{T}$, $\arg_i \max \bar{D}_i \to \arg_i \max \mu_i$ a.s. as $B \to \infty$. Or in other words the classification from the ECOC PaCT approaches the classification from just using $\arg_i \max \mu_i$ a.s.*

This leads to corollary 1 which indicates we have eliminated the main concern of a deterministic matrix.

**Corollary 1** *When the coding matrix is randomly chosen the ECOC PaCT is asymptotically Bayes Optimal ie $\arg_i \max \bar{D}_i \to \arg_i \max q_i$ a.s. as $B \to \infty$*

This theorem is a consequence of the strong law. Theorems 2 and 3 provide motivation for the ECOC procedure.

**Theorem 3** *Under assumption 1 for a randomly generated coding matrix*

$$E\bar{D}_i = E\mu_i = q_i \quad i = 1 \dots k$$

This tells us that $\bar{D}_i$ is an unbiased estimate of the conditional probability so classifying to the maximum is in a sense an unbiased estimate of the Bayes classification.
Now theorem 2 tells us that for *large B* the ECOC PaCT will be similar to classifying using $\arg_i \max \mu_i$ only. However what we mean by large depends on the rate of convergence. Theorem 4 tells us that this rate is in fact exponential.

**Theorem 4** *If we randomly choose $Z$ then, conditional on $\mathcal{T}$, for any fixed $X$*

$$Pr(\arg_i \max \bar{D}_i \neq \arg_i \max \mu_i) \leq (k - 1) \cdot e^{-mB}$$

*for some constant $m$.*

Note that theorem 4 does not depend on assumption 1. This tells us that the error rate for the ECOC PaCT is equal to the error rate using $\arg_i \max \mu_i$ plus a term which decreases exponentially in the limit. This result can be proved using Hoeffding's inequality (Hoeffding (1963)).
Of course this only gives an upper bound on the error rate and does not necessarily indicate the behavior for smaller values of $B$. Under certain conditions a Taylor expansion indicates that $Pr(\arg_i \max \bar{D}_i \neq \arg_i \max \mu_i) \approx 0.5 - m\sqrt{B}$ for small values of $m\sqrt{B}$. So we

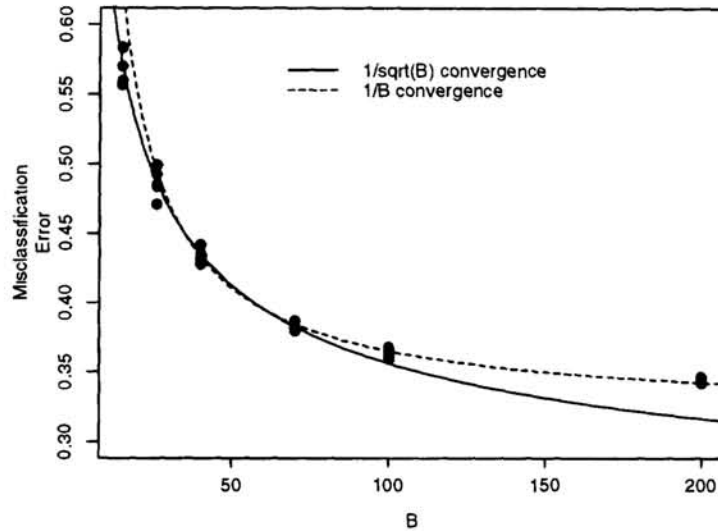

Figure 1: Best fit curves for rates $1/\sqrt{B}$ and $1/B$

might expect that for smaller values of $B$ the error rate decreases as some power of $B$ but that as $B$ increases the change looks more and more exponential.

To test this hypothesis we calculated the error rates for 6 different values of $B$ $(15, 26, 40, 70, 100, 200)$ on the LETTER data set (available from the Irvine Repository of machine learning). Each value of $B$ contains 5 points corresponding to 5 random matrices. Each point is the average over 20 random training sets. Figure 1 illustrates the results. Here we have two curves. The lower curve is the best fit of $1/\sqrt{B}$ to the first four groups. It fits those groups well but under predicts errors for the last two groups. The upper curve is the best fit of $1/B$ to the last four groups. It fits those groups well but over predicts errors for the first two groups. This supports our hypothesis that the error rate is moving through the powers of $B$ towards an exponential fit.

We can see from the figure that even for relatively low values of $B$ the reduction in error rate has slowed substantially. This indicates that almost all the remaining errors are a result of the error rate of $\arg_i \max \mu_i$ which we can not reduce by changing the coding matrix.

The coding matrix can be viewed as a method for sampling from the distribution of $1 - 2|\hat{p}_j - Z_{ij}|$. If we sample randomly we will estimate $\mu_i$ (its mean). It is well known that the optimal way to estimate such a parameter is by random sampling so it is not possible to improve on this by *designing* the coding matrix. Of course it may be possible to improve on $\arg_i \max \mu_i$ by using the training data to influence the sampling procedure and hence estimating a different quantity. However a designed coding matrix does not use the training data. It should not be possible to improve on random sampling by using such a procedure (as has been attempted in the past).

## 3   Why does the ECOC PaCT work?

The easiest way to motivate why the ECOC PaCT works, in the case of tree classifiers, is to consider a very similar method which we call the Substitution PaCT. We will show that under certain conditions the ECOC PaCT is very similar to the Substitution PaCT and then motivate the success of the later.

### 3.1 Substitution PaCT

The Substitution PaCT uses a coding matrix to form many different trees just as the ECOC PaCT does. However, instead of using the transformed training data to form a probability estimate for each two class problem, we now plug the original (ie k-class) training data back into the new tree. We use this training data to form probability estimates and classifications just as we would with a regular tree. The only difference is in how the tree is formed. Therefore, unlike the ECOC PaCT, each tree will produce a probability estimate for each of the k classes. For each class we simply average the probability estimate for that class over our $B$ trees. So if $p_i^S$ is the probability estimate for the Substitution PaCT, then

$$p_i^S = \frac{1}{B} \sum_{j=1}^{B} p_{ij} \tag{2}$$

where $p_{ij}$ is the probability estimate for the $i$th group for the tree formed from the $j$th column of the coding matrix.

Theorem 5 shows that under certain conditions the ECOC PaCT can be thought of as an approximation to the Substitution PaCT.

**Theorem 5** *Suppose that $p_{ij}$ is independent from the jth column of the coding matrix, for all i and j. Then as B approaches infinity the ECOC PaCT and Substitution PaCT will converge ie they will give identical classification rules.*

The theorem depends on an unrealistic assumption. However, empirically it is well known that trees are unstable and a small change in the data set can cause a large change in the structure of the tree so it may be reasonable to suppose that there is a low correlation.

To test this empirically we ran the ECOC and Substitution PaCT's on a simulated data set. The data set was composed of 26 classes. Each class was distributed as a bivariate normal with identity covariance matrix and uniformly distributed means. The training data consisted of 10 observations from each group. Figure 2 shows a plot of the estimated probabilities for each of the 26 classes and 1040 test data points averaged over 10 training data sets. Only points where the true posterior probability is greater than 0.01 have been plotted since groups with insignificant probabilities are unlikely to affect the classification. If the two groups were producing identical estimates we would expect the data points to lie on the dotted 45 degree line. Clearly this is not the case. The Substitution PaCT is systematically shrinking the probability estimates. However there is a very clear linear relationship ($R^2 \approx 95\%$) and since we are only interested in the arg max for each test point we might expect similar classifications. In fact this is the case with fewer than 4% of points correctly classified by one group but not the other.

### 3.2 Why does the Substitution PaCT work?

The fact that $p_i^S$ is an average of probability estimates suggests that a reduction in variability may be an explanation for the success of the Substitution PaCT. Unfortunately it has been well shown (see for example Friedman (1996)) that a reduction in variance of the probability estimates does not necessarily correspond to a reduction in the error rate. However theorem 6 provides simplifying assumptions under which a relationship between the two quantities exists.

**Theorem 6** *Suppose that*

$$p_i^T = \alpha_0^T + \alpha_1^T q_i + \sigma_T \epsilon_i^T \quad (\alpha_1^T > 0) \tag{3}$$

$$and \quad p_i^S = \alpha_0^S + \alpha_1^S q_i + \sigma_S \epsilon_i^S \quad (\alpha_1^S > 0) \tag{4}$$

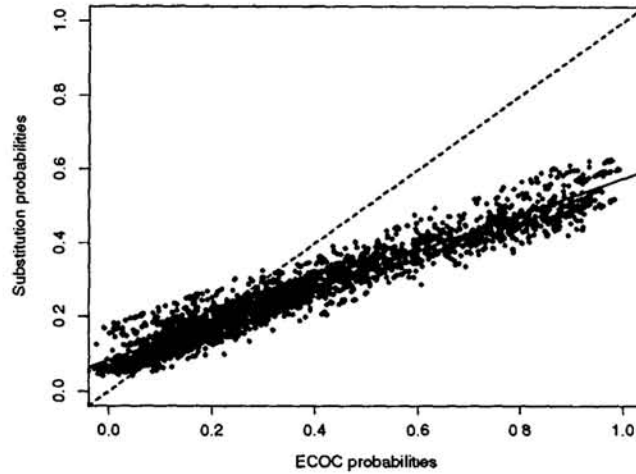

Figure 2: Probability estimates from both the ECOC and Substitution PaCT's

where $\epsilon^S$ and $\epsilon^T$ have identical joint distributions with variance 1. $p_i^T$ is the probability estimate of the ith group for a k class tree method, $\alpha_0$ and $\alpha_1$ are constants and $q_i$ is the true posterior probability. Let

$$\gamma = \frac{Var(p_i^T/\alpha_1^T)}{Var(p_{i1}/\alpha_1^S)}$$

and $\rho = corr(p_{i1}, p_{i2})$ (assumed constant for all i). Then

$$Pr(\arg\max p_i^S = \arg\max q_i) \geq Pr(\arg\max p_i^T = \arg\max q_i) \qquad (5)$$

if

$$\rho < \gamma \qquad (6)$$

and

$$B \geq \frac{1-\rho}{\gamma - \rho} \qquad (7)$$

The theorem states that under fairly general conditions, the probability that the Substitution PaCT gives the same classification as the Bayes classifier is at least as great as that for the tree method provided that the standardized variability is low enough. It should be noted that only in the case of two groups is there a direct correspondence between the error rate and 5. The inequality in 5 is strict for most common distributions (e.g. normal, uniform, exponential and gamma) of $\epsilon$.

Now there is reason to believe that in general $\rho$ will be small. This is a result of the empirical variability of tree classifiers. A small change in the training set can cause a large change in the structure of the tree and also the final probability estimates. So by changing the super group coding we might expect a probability estimate that is fairly unrelated to previous estimates and hence a low correlation.

To test the accuracy of this theory we examined the results from the simulation performed in section 3.1. We wished to estimate $\gamma$ and $\rho$. The following table summarizes our estimates for the variance and standardizing ($\alpha_1$) terms from the simulated data set.

| Classifier | $Var(p_i)$ | $\alpha_1$ | $Var(p_i/\alpha_1)$ |
|---|---|---|---|
| Substitution PaCT | 0.0515 | 0.3558 | 0.4068 |
| Tree Method | 0.0626 | 0.8225 | 0.0925 |

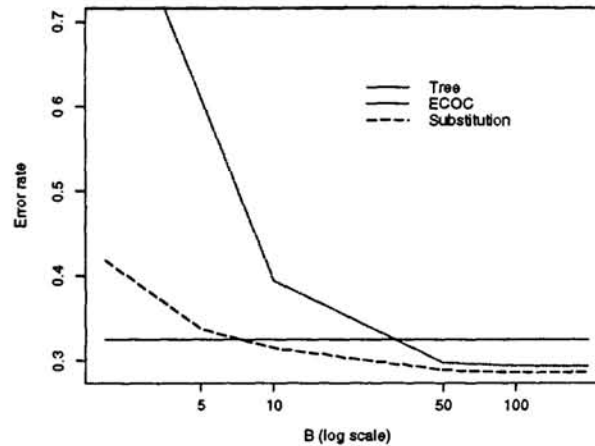

Figure 3: Error rates on the simulated data set for tree method, Substitution PaCT and ECOC PaCT plotted against $B$ (on log scale)

These quantities give us an estimate for $\gamma$ of $\hat{\gamma} = 0.227$ We also derived an estimate for $\rho$ of $\hat{\rho} = 0.125$
We see that $\rho$ is less than $\gamma$ so provided $B \geq \frac{1-\hat{\rho}}{\hat{\gamma}-\hat{\rho}} \approx 8.6$ we should see an improvement in the Substitution PaCT over a $k$ class tree classifier. Figure 3 shows that the Substitution error rate drops below that of the tree classifier at almost exactly this point.

## 4   Conclusion

The ECOC PaCT was originally envisioned as an adaption of error coding ideas to classification problems. Our results indicate that the error coding matrix is simply a method for randomly sampling from a fixed distribution. This idea is very similar to the Bootstrap where we randomly sample from the empirical distribution for a fixed data set. There you are trying to estimate the variability of some parameter. Your estimate will have two sources of error, randomness caused by sampling from the empirical distribution and the randomness from the data set itself. In our case we have the same two sources of error, error caused by sampling from $1 - 2|\hat{p}_j - Z_{ij}|$ to estimate $\mu_i$ and error's caused by $\mu$ itself. In both cases the first sort of error will reduce rapidly and it is the second type we are really interested in. It is possible to motivate the reduction in error rate of using $\arg_i \max \mu_i$ in terms of a decrease in variability, provided $B$ is large enough and our correlation ($\rho$) is small enough.

## References

Dietterich, T.G. and Bakiri G. (1995) Solving Multiclass Learning Problems via Error-Correcting Output Codes, Journal of Artificial Intelligence Research 2 (1995) 263-286
Dietterich, T. G. and Kong, E. B. (1995) Error-Correcting Output Coding Corrects Bias and Variance, Proceedings of the 12th International Conference on Machine Learning pp. 313-321 Morgan Kaufmann
Friedman, J.H. (1996) On Bias, Variance, 0/1-loss, and the Curse of Dimensionality, Dept. of Statistics, Stanford University, Technical Report
Hoeffding, W. (1963) Probability Inequalities for Sums of Bounded Random Variables. "Journal of the American Statistical Association", March, 1963